# Learning to Rank by Optimizing NDCG Measure

**Hamed Valizadegan**      **Rong Jin**
Computer Science and Engineering
Michigan State University
East Lansing, MI 48824
{valizade,rongjin}@cse.msu.edu

**Ruofei Zhang**      **Jianchang Mao**
Advertising Sciences, Yahoo! Labs
4401 Great America Parkway,
Santa Clara, CA 95054
{rzhang,jmao}@yahoo-inc.com

## Abstract

Learning to rank is a relatively new field of study, aiming to learn a ranking function from a set of training data with relevancy labels. The ranking algorithms are often evaluated using information retrieval measures, such as Normalized Discounted Cumulative Gain (NDCG) [1] and Mean Average Precision (MAP) [2]. Until recently, most learning to rank algorithms were not using a loss function related to the above mentioned evaluation measures. The main difficulty in direct optimization of these measures is that they depend on the ranks of documents, not the numerical values output by the ranking function. We propose a probabilistic framework that addresses this challenge by optimizing the expectation of NDCG over all the possible permutations of documents. A relaxation strategy is used to approximate the average of NDCG over the space of permutation, and a bound optimization approach is proposed to make the computation efficient. Extensive experiments show that the proposed algorithm outperforms state-of-the-art ranking algorithms on several benchmark data sets.

## 1  Introduction

Learning to rank has attracted the focus of many machine learning researchers in the last decade because of its growing application in the areas like information retrieval (IR) and recommender systems. In the simplest form, the so-called pointwise approaches, ranking can be treated as classification or regression by learning the numeric rank value of documents as an absolute quantity [3, 4]. The second group of algorithms, the pairwise approaches, considers the pair of documents as independent variables and learns a classification (regression) model to correctly order the training pairs [5, 6, 7, 8, 9, 10, 11]. The main problem with these approaches is that their loss functions are related to individual documents while most evaluation metrics of information retrieval measure the ranking quality for individual queries, not documents.

This mismatch has motivated the so called listwise approaches for information ranking, which treats each ranking list of documents for a query as a training instance [2, 12, 13, 14, 15, 16, 17]. Unlike the pointwise or pairwise approaches, the listwise approaches aim to optimize the evaluation metrics such as NDCG and MAP. The main difficulty in optimizing these evaluation metrics is that they are dependent on the rank position of documents induced by the ranking function, not the numerical values output by the ranking function. In the past studies, this problem was addressed either by the convex surrogate of the IR metrics or by heuristic optimization methods such as genetic algorithm. In this work, we address this challenge by a probabilistic framework that optimizes the expectation of NDCG over all the possible permutation of documents. To handle the computational difficulty, we present a relaxation strategy that approximates the expectation of NDCG in the space of permutation, and a bound optimization algorithm [18] for efficient optimization. Our experiment with several benchmark data sets shows that our method performs better than several state-of-the-art ranking techniques.

The rest of this paper is organized as follows. The related work is presented in Section 2. The proposed framework and optimization strategy is presented in Section 3. We report our experimental study in Section 4 and conclude this work in Section 5.

## 2 Related Work

We focus on reviewing the listwise approaches that are closely related to the theme of this work. The listwise approaches can be classified into two categories. The first group of approaches directly optimizes the IR evaluation metrics. Most IR evaluation metrics, however, depend on the sorted order of documents, and are non-convex in the target ranking function. To avoid the computational difficulty, these approaches either approximate the metrics with some convex functions or deploy methods (e.g., genetic algorithm [19]) for non-convex optimization. In [13], the authors introduced LambdaRank that addresses the difficulty in optimizing IR metrics by defining a virtual gradient on each document after the sorting. While [13] provided a simple test to determine if there exists an implicit cost function for the virtual gradient, theoretical justification for the relation between the implicit cost function and the IR evaluation metric is incomplete. This may partially explain why LambdaRank performs very poor when compared to MCRank [3], a simple adjustment of classification for ranking (a pointwise approach). The authors of MCRank paper even claimed that a boosting model for regression produces better results than LambdaRank. Volkovs and Zemel [17] proposed optimizing the expectation of IR measures to overcome the sorting problem, similar to the approach taken in this paper. However they use monte carlo sampling to address the intractable task of computing the expectation in the permutation space which could be a bad approximation for the queries with large number of documents. AdaRank [20] uses boosting to optimize NDCG, similar to our optimization strategy. However they deploy heuristics to embed the IR evaluation metrics in computing the weights of queries and the importance of weak rankers; i.e. it uses NDCG value of each query in the current iteration as the weight for that query in constructing the weak ranker (the documents of each query have similar weight). This is unlike our approach that the contribution of each single document to the final NDCG score is considered. Moreover, unlike our method, the convergence of AdaRank is conditional and not guaranteed. Sun et al. [21] reduced the ranking, as measured by NDCG, to pairwise classification and applied alternating optimization strategy to address the sorting problem by fixing the rank position in getting the derivative. SVM-MAP [2] relaxes the MAP metric by incorporating it into the constrains of SVM. Since SVM-MAP is designed to optimize MAP, it only considers the binary relevancy and cannot be applied to the data sets that have more than two levels of relevance judgements.

The second group of listwise algorithms defines a listwise loss function as an indirect way to optimize the IR evaluation metrics. RankCosine [12] uses cosine similarity between the ranking list and the ground truth as a query level loss function. ListNet [14] adopts the KL divergence for loss function by defining a probabilistic distribution in the space of permutation for learning to rank. FRank [9] uses a new loss function called fidelity loss on the probability framework introduced in ListNet. ListMLE [15] employs the likelihood loss as the surrogate for the IR evaluation metrics. The main problem with this group of approaches is that the connection between the listwise loss function and the targeted IR evaluation metric is unclear, and therefore optimizing the listwise loss function may not necessarily result in the optimization of the IR metrics.

## 3 Optimizing NDCG Measure

### 3.1 Notation

Assume that we have a collection of $n$ queries for training, denoted by $\mathcal{Q} = \{q^1, \ldots, q^n\}$. For each query $q^k$, we have a collection of $m_k$ documents $\mathcal{D}^k = \{d_i^k, i = 1, \ldots, m_k\}$, whose relevance to $q^k$ is given by a vector $r^k = (r_1^k, \ldots, r_{m_k}^k) \in \mathbb{Z}^{m_k}$. We denote by $F(d, q)$ the ranking function that takes a document-query pair $(d, q)$ and outputs a real number score, and by $j_i^k$ the rank of document $d_i^k$ within the collection $\mathcal{D}^k$ for query $q^k$. The NDCG value for ranking function $F(d, q)$ is then computed as following:

$$\mathcal{L}(Q, F) = \frac{1}{n} \sum_{k=1}^{n} \frac{1}{Z_k} \sum_{i=1}^{m_k} \frac{2^{r_i^k} - 1}{\log(1 + j_i^k)} \qquad (1)$$

where $Z_k$ is the normalization factor [1]. NDCG is usually truncated at a particular rank level (e.g. the first 10 retrieved documents) to emphasize the importance of the first retrieved documents.

## 3.2 A Probabilistic Framework

One of the main challenges faced by optimizing the NDCG metric defined in Equation (1) is that the dependence of document ranks (i.e., $j_i^k$) on the ranking function $F(d, q)$ is not explicitly expressed, which makes it computationally challenging. To address this problem, we consider the expectation of $\mathcal{L}(Q, F)$ over all the possible rankings induced by the ranking function $F(d, q)$, i.e.,

$$\bar{\mathcal{L}}(Q, F) = \frac{1}{n} \sum_{k=1}^{n} \frac{1}{Z_k} \sum_{i=1}^{m_k} \left\langle \frac{2^{r_i^k} - 1}{\log(1 + j_i^k)} \right\rangle_F = \frac{1}{n} \sum_{k=1}^{n} \frac{1}{Z_k} \sum_{i=1}^{m_k} \sum_{\pi^k \in S_{m_k}} \Pr(\pi^k | F, q^k) \frac{2^{r_i^k} - 1}{\log(1 + \pi^k(i))} \quad (2)$$

where $S_{m_k}$ stands for the group of permutations of $m_k$ documents, and $\pi^k$ is an instance of permutation (or ranking). Notation $\pi^k(i)$ stands for the rank position of the $i$th document by $\pi^k$. To this end, we first utilize the result in the following lemma to approximate the expectation of $1/\log(1 + \pi^k(i))$ by the expectation of $\pi^k(i)$.

**Lemma 1.** *For any distribution* $\Pr(\pi | F, q)$, *the inequality* $\bar{\mathcal{L}}(Q, F) \geq \bar{\mathcal{H}}(Q, F)$ *holds where*

$$\bar{\mathcal{H}}(Q, F) = \frac{1}{n} \sum_{k=1}^{n} \frac{1}{Z_k} \sum_{i=1}^{m_k} \frac{2^{r_i^k} - 1}{\log(1 + \langle \pi^k(i) \rangle_F)} \quad (3)$$

*Proof.* The proof follows from the fact that (a) $1/x$ is a convex function when $x > 0$ and therefore $\langle 1/\log(1 + x) \rangle \geq 1/\langle \log(1 + x) \rangle$; (b) $\log(1 + x)$ is a concave function, and therefore $\langle \log(1 + x) \rangle \leq \log(1 + \langle x \rangle)$. Combining these two factors together, we have the result stated in the lemma. □

Given $\bar{\mathcal{H}}(Q, F)$ provides a lower bound for $\bar{\mathcal{L}}(Q, F)$, in order to maximize $\bar{\mathcal{L}}(Q, F)$, we could alternatively maximize $\bar{\mathcal{H}}(Q, F)$, which is substantially simpler than $\bar{\mathcal{L}}(Q, F)$. In the next step of simplification, we rewrite $\pi^k(i)$ as

$$\pi^k(i) = 1 + \sum_{j=1}^{m_k} I(\pi^k(i) > \pi^k(j)) \quad (4)$$

where $I(x)$ outputs 1 when $x$ is true and zero otherwise. Hence, $\langle \pi^k(i) \rangle$ is written as

$$\langle \pi^k(i) \rangle = 1 + \sum_{j=1}^{m_k} \langle I(\pi^k(i) > \pi^k(j)) \rangle = 1 + \sum_{j=1}^{m_k} \Pr(\pi^k(i) > \pi^k(j)) \quad (5)$$

As a result, to optimize $\bar{\mathcal{H}}(Q, F)$, we only need to define $\Pr(\pi^k(i) > \pi^k(j))$, i.e., the marginal distribution for document $d_j^k$ to be ranked before document $d_i^k$. In the next section, we will discuss how to define a probability model for $\Pr(\pi^k | F, q^k)$, and derive pairwise ranking probability $\Pr(\pi^k(i) > \pi^k(j))$ from distribution $\Pr(\pi^k | F, q^k)$.

## 3.3 Objective Function

We model $\Pr(\pi^k | F, q^k)$ as follows

$$\Pr(\pi^k | F, q^k) = \frac{1}{Z(F, q^k)} \exp\left( \sum_{i=1}^{m_k} \sum_{j:\pi^k(j) > \pi^k(i)} (F(d_i^k, q^k) - F(d_j^k, q^k)) \right)$$

$$= \frac{1}{Z(F, q^k)} \exp\left( \sum_{i=1}^{m_k} (m_k - 2\pi^k(i) + 1) F(d_i^k, q^k) \right) \quad (6)$$

where $Z(F, q^k)$ is the partition function that ensures the sum of probability is one. Equation (6) models each pair $(d_i^k, d_j^k)$ of the ranking list $\pi^k$ by the factor $\exp(F(d_i^k, q^k) - F(d_j^k, q^k))$ if $d_i^k$ is ranked before $d_j^k$ (i.e., $\pi^k(d_i^k) < \pi^k(d_j^k)$) and vice versa. This modeling choice is consistent with the idea of ranking the documents with largest scores first; intuitively, the more documents in a permutation are in the decreasing order of score, the bigger the probability of the permutation is. Using Equation (6) for $\Pr(\pi^k | F, q^k)$, we have $\bar{\mathcal{H}}(Q, F)$ expressed in terms of ranking function $F$. By maximizing $\bar{\mathcal{H}}(Q, F)$ over $F$, we could find the optimal solution for ranking function $F$.

As indicated by Equation (5), we only need to compute the marginal distribution $\Pr(\pi^k(i) > \pi^k(j))$. To approximate $\Pr(\pi^k(i) > \pi^k(j))$, we divide the group of permutation $S_{m_k}$ into two sets:

$G_a^k(i,j) = \{\pi^k | \pi^k(i) > \pi^k(j)\}$ and $G_b^k(i,j) = \{\pi^k | \pi^k(i) < \pi^k(j)\}$. Notice that there is a one-to-one mapping between these two sets; namely for any ranking $\pi^k \in G_a^k(i,j)$, we could create a corresponding ranking $\pi^k \in G_b^k(i,j)$ by switching the rankings of document $d_i^k$ and $d_j^k$ and vice versa. The following lemma allows us to bound the marginal distribution $\Pr(\pi^k(i) > \pi^k(j))$.

**Lemma 2.** *If $F(d_i^k, q^k) > F(d_j^k, q^k)$, we have*

$$\Pr(\pi^k(i) > \pi^k(j)) \leq \frac{1}{1 + \exp\left[2(F(d_i^k, q^k) - F(d_j^k, q^k))\right]} \tag{7}$$

*Proof.*

$$
\begin{aligned}
1 &= \sum_{\pi^k \in G_a^k(i,j)} \Pr(\pi^k | F, q^k) + \sum_{\pi^k \in G_b^k(i,j)} \Pr(\pi^k | F, q^k) \\
&= \sum_{\pi^k \in G_a^k(i,j)} \Pr(\pi^k | F, q^k) \left(1 + \exp\left[2(\pi^k(i) - \pi^k(j))(F(d_i^k, q^k) - F(d_j^k, q^k))\right]\right) \\
&\geq \sum_{\pi^k \in G_a^k(i,j)} \left(\Pr(\pi^k | F, q^k)\left(1 + \exp\left[2(F(d_i^k, q^k) - F(d_j^k, q^k))\right]\right)\right) \\
&= \left(1 + \exp\left[2(F(d_i^k, q^k) - F(d_j^k, q^k))\right]\right) \Pr\left(\pi^k(i) > \pi^k(j)\right)
\end{aligned}
$$

We used the definition of $\Pr(\pi^k | F, q^k)$ in Equation (6) to find $G_b^k(i,j)$ as the dual of $G_a^k(i,j)$ in the first step of the proof. The inequality in the proof is because $\pi^k(i) - \pi^k(j) \geq 1$ and the last step is because $\Pr(\pi^k | F, q^k)$ is the only term dependent on $\pi$. $\square$

This lemma indicates that we could approximate $\Pr(\pi^k(i) > \pi^k(j))$ by a simple logistic model. The idea of using logistic model for $\Pr(\pi^k(i) > \pi^k(j))$ is not new in learning to rank [7, 9]; however it has been taken for granted and no justification has been provided in using it for learning to rank. Using the logistic model approximation introduced in Lemma 2, we now have $\langle \pi^k(i) \rangle$ written as

$$\langle \pi^k(i) \rangle \approx 1 + \sum_{j=1}^{m_k} \frac{1}{1 + \exp\left[2(F(d_i^k, q^k) - F(d_j^k, q^k))\right]} \tag{8}$$

To simplify our notation, we define $F_i^k = 2F(d_i^k, q^k)$, and rewrite the above expression as

$$\langle \pi^k(i) \rangle = 1 + \sum_{j=1}^{m_k} \Pr(\pi^k(i) > \pi^k(j)) \approx 1 + \sum_{j=1}^{m_k} \frac{1}{1 + \exp(F_i^k - F_j^k)}$$

Using the above approximation for $\langle \pi^k(i) \rangle$, we have $\bar{\mathcal{H}}$ in Equation (3) written as

$$\bar{\mathcal{H}}(Q, F) \approx \frac{1}{n} \sum_{k=1}^{n} \frac{1}{Z_k} \sum_{i=1}^{m_k} \frac{2^{r_i^k} - 1}{\log(2 + A_i^k)} \tag{9}$$

where

$$A_i^k = \sum_{j=1}^{m_k} \frac{I(j \neq i)}{1 + \exp(F_i^k - F_j^k)} \tag{10}$$

We define the following proposition to further simplify the objective function:

**Proposition 1.**

$$\frac{1}{\log(2 + A_i^k)} \geq \frac{1}{\log(2)} - \frac{A_i^k}{2\left[\log(2)\right]^2}$$

The proof is due to the Taylor expansion of convex function $1/log(2 + x)$, $x > -1$ around $x = 0$ noting that $A_i^k > 0$ (the proof of convexity of $1/log(1 + x)$ is given in Lemma 1). By plugging the result of this proposition to the objective function in Equation (9), the new objective is to minimize the following quantity:

$$\bar{\mathcal{M}}(Q, F) \approx \frac{1}{n} \sum_{k=1}^{n} \frac{1}{Z_k} \sum_{i=1}^{m_k} (2^{r_i^k} - 1) A_i^k \tag{11}$$

The objective function in Equation (11) is explicitly related to $F$ via term $A_i^k$. In the next section, we aim to derive an algorithm that learns an effective ranking function by efficiently minimizing $\bar{\mathcal{M}}$. It is also important to note that although $\bar{\mathcal{M}}$ is no longer a rigorous lower bound for the original objective function $\bar{\mathcal{L}}$, our empirical study shows that this approximation is very effective in identifying the appropriate ranking function from the training data.

### 3.4 Algorithm

To minimize $\bar{\mathcal{M}}(Q, F)$ in Equation (11), we employ the bound optimization strategy [18] that iteratively updates the solution for $F$. Let $F_i^k$ denote the value obtained so far for document $d_i^k$. To improve NDCG, following the idea of Adaboost, we restrict the new ranking value for document $d_i^k$, denoted by $\tilde{F}_i^k$, is updated as to the following form:

$$\tilde{F}_i^k = F_i^k + \alpha f_i^k \tag{12}$$

where $\alpha > 0$ is the combination weight and $f_i^k = f(d_i^k, q^k) \in \{0, 1\}$ is a binary value. Note that in the above, we assume the ranking function $F(d, q)$ is updated iteratively with an addition of binary classification function $f(d, q)$, which leads to efficient computation as well as effective exploitation of the existing algorithms for data classification. . To construct a lower bound for $\bar{\mathcal{M}}(Q, F)$, we first handle the expression $[1 + \exp(F_i^k - F_j^k)]^{-1}$, summarized by the following proposition.

**Proposition 2.**

$$\frac{1}{1 + \exp(\tilde{F}_i^k - \tilde{F}_j^k)} \leq \frac{1}{1 + \exp(F_i^k - F_j^k)} + \gamma_{i,j}^k \left[ \exp(\alpha(f_j^k - f_i^k)) - 1 \right] \tag{13}$$

*where*

$$\gamma_{i,j}^k = \frac{\exp(F_i^k - F_j^k)}{\left(1 + \exp(F_i^k - F_j^k)\right)^2} \tag{14}$$

The proof of this proposition can be found in Appendix A. This proposition separates the term related to $F_i^k$ from that related to $\alpha f_i^k$ in Equation (11), and shows how the new weak ranker (i.e., the binary classification function $f(d, q)$) will affect the current ranking function $F(d, q)$. Using the above proposition, we can derive the upper bound for M (Theorem 1) as well as a closed form solution for $\alpha$ given the solution for F (Theorem 2).

**Theorem 1.** *Given the solution for binary classifier $f_i^d$, the optimal $\alpha$ that minimizes the objective function in Equation (11) is*

$$\alpha = \frac{1}{2} \log \left( \frac{\sum_{k=1}^n \sum_{i,j=1}^{m_k} \frac{2^{r_i^k} - 1}{Z_k} \theta_{i,j}^k I(f_j^k < f_i^k)}{\sum_{k=1}^n \sum_{i,j=1}^{m_k} \frac{2^{r_i^k} - 1}{Z_k} \theta_{i,j}^k I(f_j^k > f_i^k)} \right) \tag{15}$$

*where $\theta_{i,j}^k = \gamma_{i,j}^k I(j \neq i)$.*

**Theorem 2.**

$$\bar{\mathcal{M}}(Q, \tilde{F}) \leq \bar{\mathcal{M}}(Q, F) + \gamma(\alpha) + \frac{\exp(3\alpha) - 1}{3} \sum_{k=1}^n \sum_{i=1}^{m_k} f_i^k \left( \sum_{j=1}^{m_k} \frac{2^{r_i^k} - 2^{r_j^k}}{Z_k} \theta_{i,j}^k \right)$$

*where $\gamma(\alpha)$ is only a function of $\alpha$ with $\gamma(0) = 0$.*

The proofs of these theorems are provided in Appendix B and Appendix C respectively. Note that the bound provided by Theorem 2 is tight because by setting $\alpha = 0$, the inequality reduces to equality resulting $\bar{\mathcal{M}}(Q, \tilde{F}) = \bar{\mathcal{M}}(Q, F)$. The importance of this theorem is that the optimal solution for $f_i^k$ can be found without knowing the solution for $\alpha$.

Algorithm 1 [1] summarizes the procedure in minimizing the objective function in Equation (11). First, it computes $\theta_{ij}^k$ for every pair of documents of query $k$. Then, it computes $w_i^k$, a weight for each document which can be positive or negative. A positive weight $w_i^k$ indicates that the ranking position of $d_i^k$ induced by the current ranking function $F$ is less than its true rank position, while a negative weight $w_i^k$ shows that ranking position of $d_i^k$ induced by the current $F$ is greater than its true rank position. Therefore, the sign of weight $w_i^k$ provides a clear guidance for how to construct the next weak ranker, the binary classifier in our case; that is, the documents with a positive $w_i^k$ should be labeled as $+1$ by the binary classifier and those with negative $w_i^k$ should be labeled as $-1$. The magnitude of $w_i^k$ shows how much the corresponding document is misplaced in the ranking. In other words, it shows the importance of correcting the ranking position of document $d_i^k$ in terms of improving the value of NDCG. This leads to maximizing $\eta$ given in Equation (17) which can be considered as some sort of classification accuracy. We use sampling strategy in order to maximize $\eta$ because most binary classifiers do not support the weighted training set; that is, we first sample the documents according to $|w_i^k|$ and then construct a binary classifier with the sampled documents. It can be shown that the proposed algorithm reduces the objective function $M$ exponentially (the proof is removed due to the lack of space).

**Algorithm 1** NDCG_Boost: A Boosting Algorithm for Maximizing NDCG
---
1: Initialize $F(d_i^k) = 0$ for all documents
2: **repeat**
3:    Compute $\theta_{i,j}^k = \gamma_{i,j}^k I(j \neq i)$ for all document pairs of each query. $\gamma_{i,j}^k$ is given in Eq. (14).
3:    Compute the weight for each document as

$$w_i^k = \sum_{j=1}^{m_k} \frac{2^{r_i^k} - 2^{r_j^k}}{Z_k} \theta_{i,j}^k \qquad (16)$$

3:    Assign each document the following class label $y_i^k = \text{sign}(w_i^k)$.
4:    Train a classifier $f(\mathbf{x}) : \mathbb{R}^d \rightarrow \{0, 1\}$ that maximizes the following quantity

$$\eta \;=\; \sum_{k=1}^{n} \sum_{i=1}^{m_k} |w_i^k| f(d_i^k) y_i^k \qquad (17)$$

5:    Predict $f_i$ for all documents in $\{D^k, i = 1, \ldots, n\}$
6:    Compute the combination weight $\alpha$ as provided in Equation (15).
7:    Update the ranking function as $F_i^k \leftarrow F_i^k + \alpha f_i^k$.
8: **until** reach the maximum number of iterations
---

# 4 Experiments

To study the performance of NDCG_Boost we use the latest version (version 3.0) of LETOR package provided by Microsoft Research Asia [22]. LETOR Package includes several benchmark data data, baselines and evaluation tools for research on learning to rank.

## 4.1 Letor Data Sets

There are seven data sets provided in the LETOR package: OHSUMED, Top Distillation 2003 (TD2003), Top Distillation 2004 (TD2004), Homepage Finding 2003 (HP2003), Homepage Finding 2003 (HP2003), Named Page Finding 2003 (NP2003) and Named Page Finding 2004 (NP2004) [2].

There are 106 queries in the OSHUMED data sets with a number of documents for each query. The relevancy of each document in OHSUMED data set is scored 0 (irrelevant), 1 (possibly) or 2 (definitely). The total number of query-document relevancy judgments provided in OHSUMED data set is 16140 and there are 45 features for each query-document pair. For TD2003, TD2004, HP2003, HP2004 and NP2003, there are 50, 75, 75, 75 and 150 queries, respectively, with about 1000 retrieved documents for each query. This amounts to a total number of 49171, 74170, 74409, 73834 and 147606 query-document pairs for TD2003, TD2004, HP2003, HP2004 and NP2003 respectively. For these data sets, there are 63 features extracted for each query-document pair and a binary relevancy judgment for each pair is provided.

For every data sets in LETOR, five partitions are provided to conduct the five-fold cross validation, each includes training, test and validation sets. The results of a number of state-of-the-art learning to rank algorithms are also provided in the LETOR package. Since these baselines include some of the most well-known learning to rank algorithms from each category (pointwise, pairwise and listwise), we use them to study the performance of NDCG_Boost. Here is the list of these baselines (the details can be found in the LETOR web page):

**Regression:** This is a simple linear regression which is a basic pointwise approach and can be considered as a reference point.

**RankSVM:** RankSVM is a pairwise approach using Support Vector Machine [5].

**FRank:** FRank is a pairwise approach. It uses similar probability model to RankNet [7] for the relative rank position of two documents, with a novel loss function called Fidelity loss function [9]. TSai et al [9] showed that FRank performs much better than RankNet.

**ListNet:** ListNet is a listwise learning to rank algorithm [14]. It uses cross-entropy loss as its listwise loss function.

**AdaRank_NDCG:** This is a listwise boosting algorithm that incorporates NDCG in computing the samples and combination weights [20].

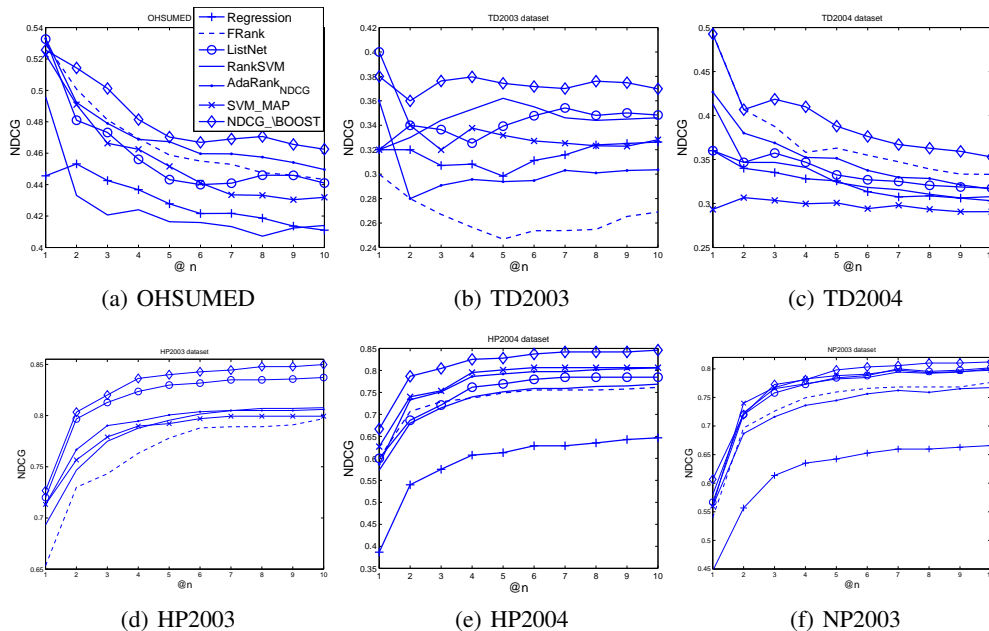

(a) OHSUMED      (b) TD2003      (c) TD2004

(d) HP2003      (e) HP2004      (f) NP2003

Figure 1: The experimental results in terms of NDCG for Letor 3.0 data sets

**SVM_MAP:** SVM_MAP is a support vector machine with MAP measure used in the constraints. It is a listwise approach [2].

While the validation set is used in finding the best set of parameters in the baselines in LETOR, it is not being used for NDCG_Boost in our experiments. For NDCG_Boost, we set the maximum number of iteration to 100 and use decision stump as the weak ranker.

Figure 1 provides the the average results of five folds for different learning to rank algorithms in terms of NDCG @ each of the first 10 truncation level on the LETOR data sets [3]. Notice that the performance of algorithms in comparison varies from one data set to another; however NDCG_Boost performs almost always the best. We would like to point out a few statistics; On OHSUMED data set, NDCG_Boost performs $0.50$ at $NDCG@3$, a $4\%$ increase in performance, compared to FRANK, the second best algorithm. On TD2003 data set, this value for NDCG_Boost is $0.375$ that shows a $10\%$ increase, compared with RankSVM ($0.34$), the second best method. On HP2004 data set, NDCG_Boost performs $0.80$ at $NDCG@3$, compared to $0.75$ of SVM_MAP, the second best method, which indicates a $6\%$ increase. Moreover, among all the methods in comparison, NDCG_Boost appears to be the most stable method across all the data sets. For example, FRank, which performs well in OHSUMED and TD2004 data sets, yields a poor performance on TD2003, HP2003 and HP 2004. Similarly, AdaRank_NDCG achieves a decent performance on OHSUMED data set, but fails to deliver accurate ranking results on TD2003, HP2003 and NP2003. In fact, both AdaRank_NDCG and FRank perform even worse than the simple Regression approach on TD2003, which further indicates their instability. As another example, ListNet and RankSVM, which perform well on TD2003 are not competitive to NDCG_boost on OHSUMED and TD2004 data sets.

## 5 Conclusion

Listwise approach is a relatively new approach to learning to rank. It aims to use a query-level loss function to optimize a given IR measure. The difficulty in optimizing IR measure lies in the inherited sort function in the measure. We address this challenge by a probabilistic framework that optimizes the expectation of NDCG over all the possible permutations of documents. We present a relaxation strategy to effectively approximate the expectation of NDCG, and a bound optimization strategy for efficient optimization. Our experiments on benchmark data sets shows that our method is superior to the state-of-the-art learning to rank algorithms in terms of performance and stability.

## 6 Acknowledgements

The work was supported in part by the Yahoo! Labs[4] and National Institute of Health (1R01GM079688-01). Any opinions, findings, and conclusions or recommendations expressed in this material are those of the authors and do not necessarily reflect the views of Yahoo! and NIH.

## A Proof of Proposition 2

$$
\begin{aligned}
\frac{1}{1 + \exp(\tilde{F}_i^k - \tilde{F}_j^k)} &= \frac{1}{1 + \exp(F_i^k - F_j^k + \alpha(f_i^k - f_j^k))} \\
&= \frac{1}{1 + \exp(F_i^k - F_j^k)} \left( \frac{1}{1 + \exp(F_i^k - F_j^k)} + \frac{\exp(F_i^k - F_j^k)}{1 + \exp(F_i^k - F_j^k)} \exp(\alpha(f_i^k - f_j^k)) \right)^{-1} \\
&\leq \frac{1}{1 + \exp(F_i^k - F_j^k)} \left( 1 - \frac{\exp(F_i^k - F_j^k)}{1 + \exp(F_i^k - F_j^k)} + \frac{\exp(F_i^k - F_j^k)}{1 + \exp(F_i^k - F_j^k)} \exp(\alpha(f_j^k - f_i^k)) \right) \\
&= \frac{1}{1 + \exp(F_i^k - F_j^k)} + \gamma_{i,j}^k \left[ \exp(\alpha(f_j^k - f_i^k) - 1 \right]
\end{aligned}
$$

The first step is a simple manipulations of the terms and the second step is due to the convexity of inverse function on $R^+$.

## B Proof of Theorem 1

In order to obtain the result of the Theorem 1, we first plug Equation (13) in Equation (11). This leads to minimizing $\sum_{k=1}^{n} \sum_{i,j=1}^{m_k} \frac{2^{r_i^k} - 1}{Z_k} \theta_{i,j}^k \left[ \exp(\alpha(f_j^k - f_i^k)) \right]$, the term related to $\alpha$. Since $f_i^k$ takes binary values 0 and 1, we have the following:

$$
\sum_{k=1}^{n} \sum_{i,j=1}^{m_k} \frac{2^{r_i^k} - 1}{Z_k} \theta_{i,j}^k \exp(\alpha(f_j^k - f_i^k)) = \sum_{k=1}^{n} \sum_{i,j=1}^{m_k} \frac{2^{r_i^k} - 1}{Z_k} \theta_{i,j}^k \left( \exp(\alpha) I(f_j^k > f_i^k) + \exp(-\alpha) I(f_j^k < f_i^k) \right)
$$

Getting the partial derivative of this term respect to $\alpha$ and having it equal to zero results the theorem.

## C Proof of Theorem 2

First, we provide the following proposition to handle $\exp(\alpha(f_j^k - f_i^k))$.

**Proposition 3.** *If $x, y \in [0, 1]$, we have*

$$
\exp(\alpha(x - y)) \leq \frac{\exp(3\alpha) - 1}{3}(x - y) + \frac{\exp(3\alpha) + \exp(-3\alpha) + 1}{3} \tag{18}
$$

*Proof.* Due to the convexity of $exp$ function, we have:

$$
\begin{aligned}
\exp(\alpha(x - y)) &= \exp(3\alpha \frac{x - y + 1}{3} + 0 \times \frac{1 - x + y}{3} + \frac{1}{3} \times -3\alpha) \\
&\leq \frac{x - y + 1}{3} \exp(3\alpha) + \frac{1 - x + y}{3} + \frac{1}{3} \exp(-3\alpha)
\end{aligned}
$$

$\square$

Using the result in the above proposition, we can bound the last term in Equation (13) as follows:

$$
\theta_{i,j}^k \left[ \exp(\alpha(f_j^k - f_i^k) - 1 \right] \leq \theta_{i,j}^k \left( \frac{\exp(3\alpha) - 1}{3}(f_j^k - f_i^k) + \frac{\exp(3\alpha) + \exp(-3\alpha) - 2}{3} \right) \tag{19}
$$

Using the result in Equation (19) and (13), we have $\bar{\mathcal{M}}(Q, \tilde{F})$ in Equation (11) bounded as

$$
\begin{aligned}
\bar{\mathcal{M}}(Q, \tilde{F}) &\leq \bar{\mathcal{M}}(Q, F) + \gamma(\alpha) + \frac{\exp(3\alpha) - 1}{3} \sum_{k=1}^{n} \sum_{i=1}^{m_k} \frac{2^{r_i^k} - 1}{Z_k} \sum_{j=1}^{m_k} \theta_{i,j}^k (f_i^k - f_j^k) \\
&= \bar{\mathcal{M}}(Q, F) + \gamma(\alpha) + \frac{\exp(3\alpha) - 1}{3} \sum_{k=1}^{n} \sum_{i=1}^{m_k} f_i^k \left( \sum_{j=1}^{m_k} \frac{2^{r_i^k} - 2^{r_j^k}}{Z_k} \theta_{i,j}^k \right)
\end{aligned}
$$

## Footnotes

[1]Notice that we use $F(d_i^k)$ instead of $F(d_i^k, q^k)$ to simplify the notation in the algorithm.

[2]The experiment result for the last data set is not reported due to the lack of space.

[3]NDCG is commonly measured at the first few retrieved documents to emphasize their importance.

[4] The first author has been supported as a part-time intern in Yahoo!

# References

[1] Kalervo Järvelin and Jaana Kekäläinen. Ir evaluation methods for retrieving highly relevant documents. In *SIGIR 2000: Proceedings of the 23th annual international ACM SIGIR conference on Research and development in information retrieval*, pages 41–48, 2000.

[2] Yisong Yue, Thomas Finley, Filip Radlinski, and Thorsten Joachims. A support vector method for optimizing average precision. In *SIGIR 2007: Proceedings of the 30th annual Int. ACM SIGIR Conf. on Research and development in information retrieval*, pages 271–278, 2007.

[3] Ping Li, Christopher Burges, and Qiang Wu. Mcrank: Learning to rank using multiple classification and gradient boosting. In *Neural Information Processing System 2007*.

[4] Ramesh Nallapati. Discriminative models for information retrieval. In *SIGIR '04: Proceedings of the 27th annual international ACM SIGIR conference on Research and development in information retrieval*, pages 64–71, New York, NY, USA, 2004. ACM.

[5] Ralf Herbrich, Thore Graepel, and Klaus Obermayer. Support vector learning for ordinal regression. In *Int. Conf. on Artificial Neural Networks 1999*, pages 97–102, 1999.

[6] Yoav Freund, Raj Iyer, Robert E. Schapire, and Yoram Singer. An efficient boosting algorithm for combining preferences. *Journal of Machine Learning Research*, 4:933–969, 2003.

[7] Chris Burges, Tal Shaked, Erin Renshaw, Ari Lazier, Matt Deeds, Nicole Hamilton, and Greg Hullender. Learning to rank using gradient descent. In *International Conference on Machine Learning 2005*, 2005.

[8] Yunbo Cao, Jun Xu, Tie-Yan Liu, Hang Li, Yalou Huang, and Hsiao-Wuen Hon. Adapting ranking svm to document retrieval. In *SIGIR 2006: Proceedings of the 29th annual international ACM SIGIR conference on Research and development in information retrieval*, pages 186–193, 2006.

[9] Ming_Feng Tsai, Tie yan Liu, Tao Qin, Hsin hsi Chen, and Wei ying Ma. Frank: A ranking method with fidelity loss. In *SIGIR 2007: Proceedings of the 30th annual international ACM SIGIR conference on Research and development in information retrieval*, 2007.

[10] Rong Jin, Hamed Valizadegan, and Hang Li. Ranking refinement and its application to information retrieval. In *WWW '08: Proc. of the 17th int. conference on World Wide Web*.

[11] Steven C.H. Hoi and Rong Jin. Semi-supervised ensemble ranking. In *Proceedings of Association for the Advancement of Artificial Intelligence (AAAI2008)*.

[12] Tao Qin, Tie yan Liu, Ming feng Tsai, Xu dong Zhang, and Hang Li. Learning to search web pages with query-level loss functions. Technical report, 2006.

[13] Christopher J. C. Burges, Robert Ragno, and Quoc V. Le. Learning to rank with nonsmooth cost functions. In *Neural Information Processing System 2006*, 2006.

[14] Zhe Cao and Tie yan Liu. Learning to rank: From pairwise approach to listwise approach. In *International Conference on Machine Learning 2007*, pages 129–136, 2007.

[15] Fen Xia, Tie-Yan Liu, Jue Wang, Wensheng Zhang, and Hang Li. Listwise approach to learning to rank: theory and algorithm. In *Int. Conf. on Machine Learning 2008*, pages 1192–1199, 2008.

[16] Michael Taylor, John Guiver, Stephen Robertson, and Tom Minka. Softrank: optimizing nonsmooth rank metrics.

[17] Maksims N. Volkovs and Richard S. Zemel. Boltzrank: learning to maximize expected ranking gain. In *ICML '09: Proceedings of the 26th Annual International Conference on Machine Learning*, pages 1089–1096, New York, NY, USA, 2009. ACM.

[18] Ruslan Salakhutdinov, Sam Roweis, and Zoubin Ghahramani. On the convergence of bound optimization algorithms. In *Proc. 19th Conf. in Uncertainty in Artificial Intelligence (UAI 03)*.

[19] Jen-Yuan Yeh, Yung-Yi Lin, Hao-Ren Ke, and Wei-Pang Yang. Learning to rank for information retrieval using genetic programming. In *SIGIR 2007 workshop: Learning to Rank for Information Retrieval*.

[20] Jun Xu and Hang Li. Adarank: a boosting algorithm for information retrieval. In *SIGIR '07: Proceedings of the 30th annual international ACM SIGIR conference on Research and development in information retrieval*, pages 391–398, 2007.

[21] Zhengya Sun, Tao Qin, Qing Tao, and Jue Wang. Robust sparse rank learning for non-smooth ranking measures. In *SIGIR '09: Proceedings of the 32nd international ACM SIGIR conference on Research and development in information retrieval*, pages 259–266, New York, NY, USA, 2009. ACM.

[22] Tie-Yan Liu, Tao Qin, Jun Xu, Wenying Xiong, and Hang Li. Letor: Benchmark dataset for research on learning to rank for information retrieval.

